# A Mixed-Signal VLSI for Real-Time Generation of Edge-Based Image Vectors

**Masakazu Yagi, Hideo Yamasaki, and Tadashi Shibata\***
Department of Electronic Engineering
*Department of Frontier Informatics
The University of Tokyo
7-3-1 Hongo, Bunkyo-ku, Tokyo, 113-8656, Japan
*mgoat@dent.osaka-u.ac.jp, hideo@if.t.u-tokyo.ac.jp, shibata@ee.t.u-tokyo.ac.jp*

## Abstract

A mixed-signal image filtering VLSI has been developed aiming at real-time generation of edge-based image vectors for robust image recognition. A four-stage asynchronous median detection architecture based on analog digital mixed-signal circuits has been introduced to determine the threshold value of edge detection, the key processing parameter in vector generation. As a result, a fully seamless pipeline processing from threshold detection to edge feature map generation has been established. A prototype chip was designed in a 0.35-$\mu$m double-polysilicon three-metal-layer CMOS technology and the concept was verified by the fabricated chip. The chip generates a 64-dimension feature vector from a 64x64-pixel gray scale image every 80$\mu$sec. This is about $10^4$ times faster than the software computation, making a real-time image recognition system feasible.

## 1  Introduction

The development of human-like image recognition systems is a key issue in information technology. However, a number of algorithms developed for robust image recognition so far [1]-[3] are mostly implemented as software systems running on general-purpose computers. Since the algorithms are generally complex and include a lot of floating point operations, they are computationally too expensive to build real-time systems. Development of hardware-friendly algorithms and their direct VLSI implementation would be a promising solution for real-time response systems.

Being inspired by the biological principle that edge information is firstly detected in the visual cortex, we have developed an edge-based image representation algorithm compatible to hardware processing. In this algorithm, multiple-direction edges extracted from an original gray scale image is utilized to form a feature vector. Since the spatial distribution of principal edges is represented by a vector, it was named Projected Principal-Edge Distribution (PPED) [4],[5], or formerly called Principal Axis

Projection (PAP) [6],[7]. (The algorithm is explained later.) Since the PPED vectors very well represent the human perception of similarity among images, robust image recognition systems have been developed using PPED vectors in conjunction with the analog soft pattern classifier [4],[8], the digital VQ (Vector Quantization) processor [9], and support vector machines [10] .

The robust nature of PPED representation is demonstrated in Fig. 1, where the system was applied to cephalometric landmark identification (identifying specific anatomical landmarks on medical radiographs) as an example, one of the most important clinical practices of expert dentists in orthodontics [6],[7]. Typical X-ray images to be experienced by apprentice doctors were converted to PPED vectors and utilized as templates for vector matching. The system performance has been proven for 250 head film samples regarding the fundamental 26 landmarks [11]. Important to note is the successful detection of the landmark on the soft tissue boundary (the tip of the lower lip) shown in Fig. 1(c). Landmarks on soft tissues are very difficult to detect as compared to landmarks on hard tissues (solid bones) because only faint images are captured on radiographs. The successful detection is due to the median algorithm that determines the threshold value for edge detection.

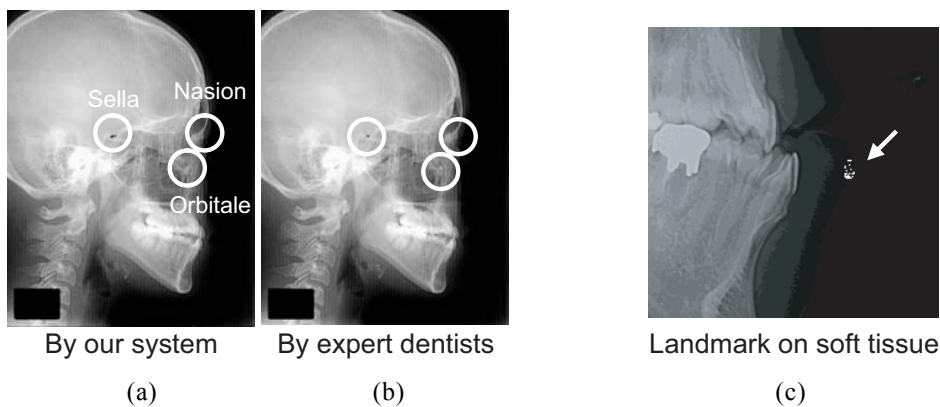

| By our system | By expert dentists | Landmark on soft tissue |
|:---:|:---:|:---:|
| (a) | (b) | (c) |

Fig. 1: Image recognition using PPED vectors: (a,b) cephalometric landmark identification; (c) successful landmark detection on soft tissue.

We have adopted the median value of spatial variance of luminance within the filtering kernel (5x5 pixels), which allows us to extract all essential features in a delicate gray scale image. However, the problem is the high computational cost in determining the median value. It takes about 0.6 sec to generate one PPED vector from a 64x64-pixel image (a standard image size for recognition in our system) on a SUN workstation, making real time processing unrealistic. About 90% of the computation time is for edge detection from an input image, in which most of the time is spent for median detection.

Then the purpose of this work is to develop a new architecture median-filter VLSI subsystem for real-time PPED-vector generation. Special attention has been paid to realize a fully seamless pipeline processing from threshold detection to edge feature map generation by employing the four-stage asynchronous median detection architecture.

## 2 Projected Principal Edge Distribution (PPED)

Projected Principal Edge Distribution (PPED) algorithm [5],[6] is briefly explained using Fig. 2(a). A 5x5-pixel block taken from a 64x64-pixel target image is subjected to edge detection filtering in four principal directions, i.e. horizontal, vertical, and ±45-degree directions. In the figure, horizontal edge filtering is shown as an example. (The filtering kernels used for edge detection are given in Fig. 2(b).) In order to determine the threshold value for edge detection, all the absolute-value differences between two neighboring pixels are calculated in both vertical and horizontal directions and the median value is taken as the threshold. By scanning the 5x5-pixel filtering kernels in the target image, four 64x64 edge-flag maps are generated, which are called feature maps. In the horizontal feature map, for example, edge flags in every four rows are accumulated and spatial distribution of edge flags are represented by a histogram having 16 elements. Similar procedures are applied to other three directions to form respective histograms each having 16 elements. Finally, a 64-dimension vector is formed by series-connecting the four histograms in the order of horizontal, +45-degree, vertical, and –45-degree.

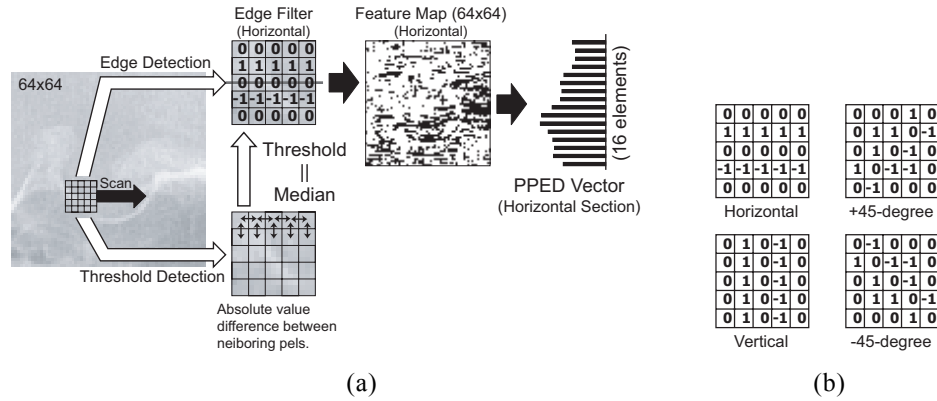

Fig. 2: PPED algorithm (a) and filtering kernels for edge detection (b).

## 3 System Organization

The system organization of the feature map generation VLSI is illustrated in Fig. 3. The system receives one column of data (8-b x 5 pixels) at each clock and stores the data in the last column of the 5x6 image buffer. The image buffer shifts all the stored data to the right at every clock. Before the edge filtering circuit (EFC) starts detecting four direction edges with respect to the center pixel in the 5x5 block, the threshold value calculated from all the pixel data in the 5x5 block must be ready in time for the processing. In order to keep the coherence of the threshold detection and the edge filtering processing, the two last-in data locating at column 5 and 6 are given to median filter circuit (MFC) in advance via absolute value circuit (AVC). AVC calculates all luminance differences between two neighboring pixels in columns 5 and 6.

In this manner, a fully seamless pipeline processing from threshold detection to edge feature map generation has been established. The key requirement here is that MFC must determine the median value of the 40 luminance difference data from the 5x5-pixel block fast enough to carry out the seamless pipeline processing. For this purpose, a four-stage asynchronous median detection architecture has been developed which is explained in the following.

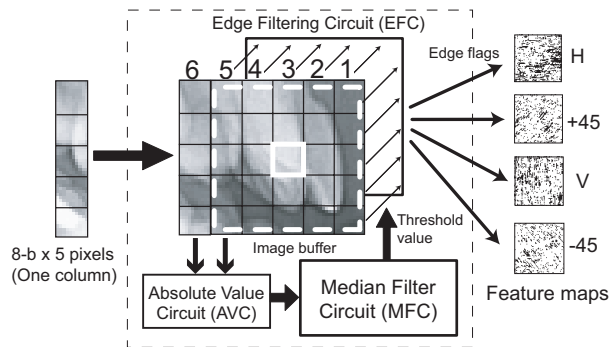

Fig. 3: System organization of feature map generation VLSI.

The well-known binary search algorithm was adopted for fast execution of median detection. The median search processing for five 4-b data is illustrated in Fig. 4 for the purpose of explanation. In the beginning, majority voting is carried out for the MSB's of all data. Namely, the number of 1's is compared with the number of 0's and the majority group wins. The majority group flag ("0" in this example) is stored as the MSB of the median value. In addition, the loser group is withdrawn in the following voting by changing all remaining bits to the loser MSB ("1" in this example). By repeating the processing, the median value is finally stored in the median value register.

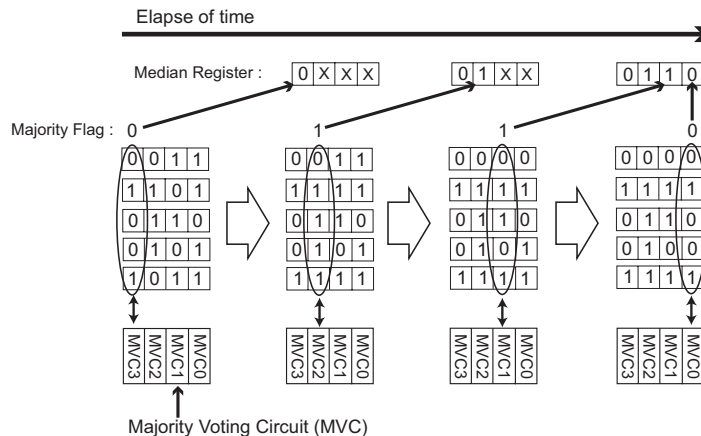

Fig. 4: Hardware algorithm for median detection by binary search.

How the median value is detected from all the 40 8-b data (20 horizontal luminance difference data and 20 vertical luminance difference data) is illustrated in Fig. 5. All the data are stored in the array of median detection units (MDU's). At each clock, the array receives four vertical luminance difference data and five horizontal luminance difference data calculated from the data in column 5 and 6 in Fig. 3. The entire data are shifted downward at each clock. The median search is carried out for the upper four bits and the lower four bits separately in order to enhance the throughput by pipelining. For this purpose, the chip is equipped with eight majority voting circuits (MVC 0~7). The upper four bits from all the data are processed by MVC 4~7 in a single clock cycle to yield the median value. In the next clock cycle, the loser information is transferred to the lower four bits within each MDU and MVC0~3 carry out the median search for the lower four bits from all the data in the array.

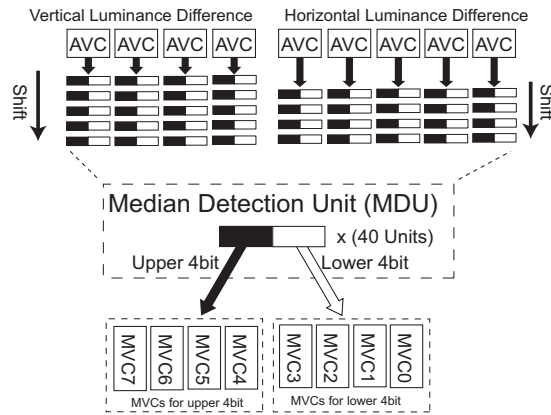

Fig. 5: Median detection architecture for all 40 luminance difference data.

The majority voting circuit (MVC) is shown in Fig. 6. Output connected CMOS inverters are employed as preamplifiers for majority detection which was first proposed in Ref. [12]. In the present implementation, however, two preamps receiving input data and inverted input data are connected to a 2-stage differential amplifier. Although this doubles the area penalty, the instability in the threshold for majority detection due to process and temperature variations has been remarkably improved as compared to the single inverter thresholding in Ref. [12]. The MVC in Fig. 6 has 41 input terminals although 40 bits of data are inputted to the circuit at one time. Bit "0" is always given to the terminal IN40 to yield "0" as the majority when there is a tie in the majority voting.

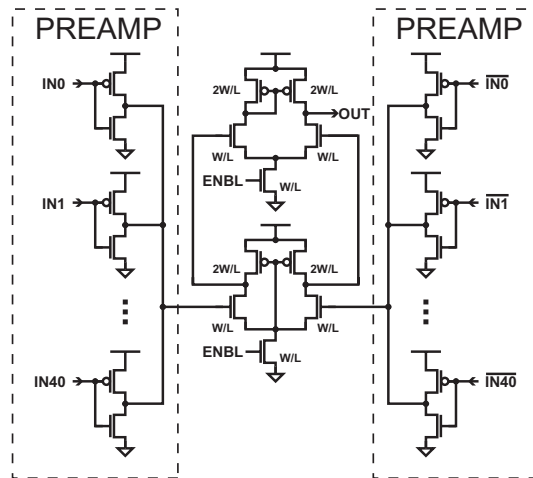

Fig. 6: Majority voting circuit (MVC).

The edge filtering circuit (EFC) in Fig. 3 is composed as a four-stage pipeline of regular CMOS digital logic. In the first two stages, four-direction edge gradients are computed, and in the succeeding two stages, the detection of the largest gradient and the thresholding is carried out to generate four edge flags.

# 4 Experimental Results

The feature map generation VLSI was fabricated in a 0.35-μm double-poly three-metal-layer CMOS technology. A photomicrograph of the proof-of-concept chip is shown in Fig. 7. The measured waveforms of the MVC at operating frequencies of 10MHz and 90MHz are demonstrated in Fig. 8. The input condition is in the worst case. Namely, 21 "1" bits and 20 "0" bits were fed to the inputs. The observed computation time is about 12 nsec which is larger than the simulation result of 2.5 nsec. This was caused by the capacitance loading due to the probing of the test circuit. In the real circuit without external probing, we confirmed the average computation time of 4~5 nsec.

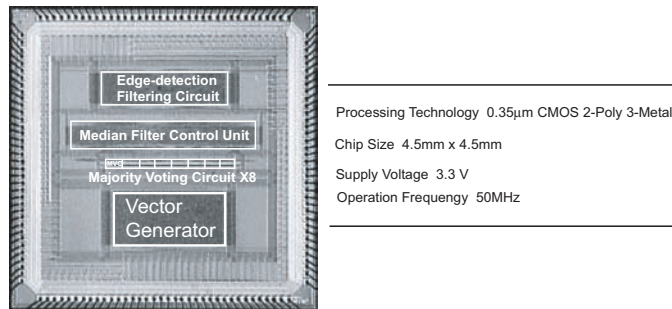

Fig. 7: Photomicrograph and specification of the fabricated proof-of-concept chip.

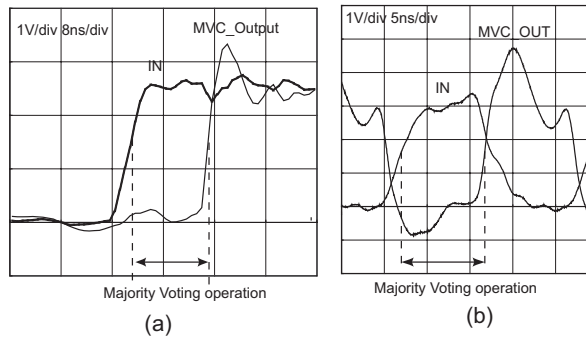

Fig. 8: Measured waveforms of majority voting circuit (MVC) at operation frequencies of 10MHz (a) and 90 MHz (b) for the worst-case input data.

The feature maps generated by the chip at the operation frequency of 25 MHz are demonstrated in Fig. 9. The power dissipation was 224 mW. The difference between the flag bits detected by the chip and those obtained by computer simulation are also shown in the figure. The number of error flags was from 80 to 120 out of 16,384 flags, only a 0.6% of the total. The occurrence of such error bits is anticipated since we employed analog circuits for median detection. However, such error does not cause any serious problems in the PPED algorithm as demonstrated in Figs. 10 and 11.

The template matching results with the top five PPED vector candidates in Sella identification are demonstrated in Fig. 11, where Manhattan distance was adopted as the dissimilarity measure. The error in the feature map generation processing yields a constant bias to the dissimilarity and does not affect the result of the maximum likelihood search.

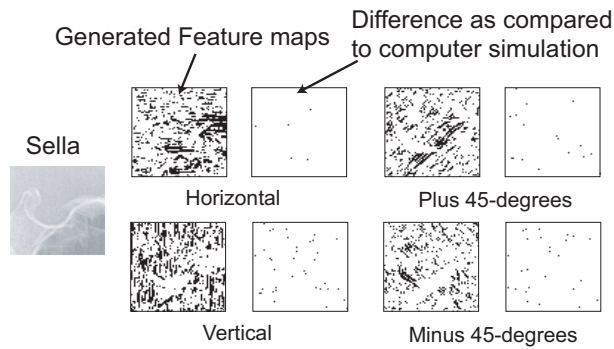

Fig. 9: Feature maps for Sella pattern generated by the chip.

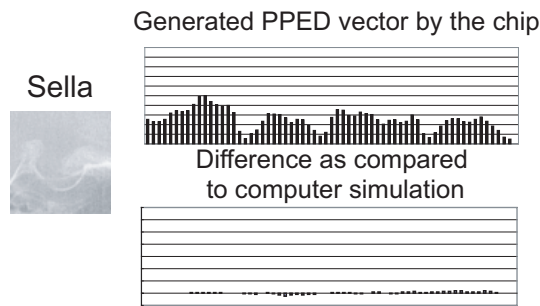

Fig. 10: PPED vector for Sella pattern generated by the chip. The difference in the vector components between the PPED vector generated by the chip and that obtained by computer simulation is also shown.

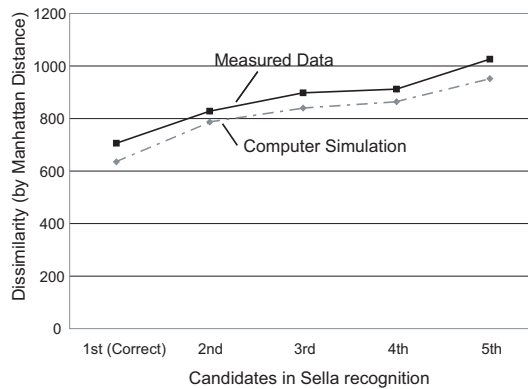

Fig. 11: Comparison of template matching results.

## 5 Conclusion

A mixed-signal median filter VLSI circuit for PPED vector generation is presented. A four-stage asynchronous median detection architecture based on analog digital mixed-signal circuits has been introduced. As a result, a fully seamless pipeline processing from threshold detection to edge feature map generation has been established. A prototype chip was designed in a 0.35-μm CMOS technology and the fab-

ricated chip generates an edge based image vector every 80 μsec, which is about $10^4$ times faster than the software computation.

## Acknowledgments

The VLSI chip in this study was fabricated in the chip fabrication program of VLSI Design and Education Center (VDEC), the University of Tokyo with the collaboration by Rohm Corporation and Toppan Printing Corporation. The work is partially supported by the Ministry of Education, Science, Sports, and Culture under Grant-in-Aid for Scientific Research (No. 14205043) and by JST in the program of CREST.

## References

[1] C. Liu and Harry Wechsler, "Gabor feature based classification using the enhanced fisher linear discriminant model for face recognition", *IEEE Transactions on Image Processing*, Vol. 11, No.4, Apr. 2002.

[2] C. Yen-ting, C. Kuo-sheng, and L. Ja-kuang, "Improving cephalogram analysis through feature subimage extraction", *IEEE Engineering in Medicine and Biology Magazine*, Vol. 18, No. 1, 1999, pp. 25-31.

[3] H. Potlapalli and R. C. Luo, "Fractal-based classification of natural textures", *IEEE Transactions on Industrial Electronics*, Vol. 45, No. 1, Feb. 1998.

[4] T. Yamasaki and T. Shibata, "Analog Soft-Pattern-Matching Classifier Using Floating-Gate MOS Technology," Advances in Neural Information Processing Systems 14, Vol. II, pp. 1131-1138.

[5] Masakazu Yagi, Tadashi Shibata, "An Image Representation Algorithm Compatible to Neural-Associative-Processor-Based Hardware Recognition Systems," IEEE Trans. Neural Networks, Vol. 14, No. 5, pp. 1144-1161, September (2003).

[6] M. Yagi, M. Adachi, and T. Shibata, "A hardware-friendly soft-computing algorithm for image recognition," in Proc. EUSIPCO 2000, Sept. 2000, pp. 729-732.

[7] M. Yagi, T. Shibata, and K. Takada, "Human-perception-like image recognition system based on the associative processor architecture," in Proc. EUSIPCO 2002, Vol. I, pp. 103-106, Sept. 2002.

[8] M. Yagi and T. Shibata, "An associative-processor-based mixed signal system for robust image recognition," in Proc. ISCAS 2002, May 2002, pp. V-137-V-140.

[9] M. Ogawa, K. Ito, and T. Shibata, "A general-purpose vector-quantization processor employing two-dimensional bit-propagating winner-take-all," in Symp. on VLSI Circuits Dig. Tech. Papers, Jun. 2002, p.p. 244-247.

[10] S. Chakrabartty, M. Yagi, T. Shibata, and G. Cauwenberghs, "Robust Cephalometric Landmark Identification Using Support Vector Machines," ICASSP 2003, Hong Kong, April 6-10, 2003, pp. II-825-II-828.

[11] Masakazu Yagi, Tadashi Shibata, Chihiro Tanikawa, and Kenji Takada, "A Robust Medical Image Recognition System Employing Edge-Based Feature Vector Representation," in the Proceeding of 13th Scandinavian Conference on Image Analysis (SCIA2003), pp.534-540, Goteborg, Sweden, Jun. 29-Jul. 2, 2003.

[12] C.L. Lee and C.-W. Jen, "Bit-sliced median filter design based on majority gate," in IEE Proceedings-G, Vol. 139, No. 1, Feb. 1992, pp. 63-71.
